# Finding Exemplars from Pairwise Dissimilarities via Simultaneous Sparse Recovery

**Ehsan Elhamifar**
EECS Department
University of California, Berkeley

**Guillermo Sapiro**
ECE, CS Department
Duke University

**René Vidal**
Center for Imaging Science
Johns Hopkins University

## Abstract

Given pairwise dissimilarities between data points, we consider the problem of finding a subset of data points, called *representatives* or *exemplars*, that can efficiently describe the data collection. We formulate the problem as a row-sparsity regularized trace minimization problem that can be solved efficiently using convex programming. The solution of the proposed optimization program finds the representatives and the probability that each data point is associated with each one of the representatives. We obtain the range of the regularization parameter for which the solution of the proposed optimization program changes from selecting one representative for all data points to selecting all data points as representatives. When data points are distributed around multiple clusters according to the dissimilarities, we show that the data points in each cluster select representatives only from that cluster. Unlike metric-based methods, our algorithm can be applied to dissimilarities that are asymmetric or violate the triangle inequality, i.e., it does not require that the pairwise dissimilarities come from a metric. We demonstrate the effectiveness of the proposed algorithm on synthetic data as well as real-world image and text data.

## 1 Introduction

Finding a subset of data points, called *representatives* or *exemplars*, which can efficiently describe the data collection, is an important problem in scientific data analysis with applications in machine learning, computer vision, information retrieval, etc. Representatives help to summarize and visualize datasets of images, videos, text and web documents. Computational time and memory requirements of classification algorithms improve by working on representatives, which contain much of the information of the original data collection. For example, the efficiency of the NN method improves [1] by comparing tests samples to $K$ representatives as opposed to all $N$ training samples, where typically we have $K \ll N$. Representatives provide clustering of data points, and, as the most prototypical data points, can be used for efficient synthesis/generation of new data points.

The problem of finding representative data has been well-studied in the literature [2, 3, 4, 5, 6, 7, 8]. Depending on the type of the information that should be preserved by the representatives, algorithms can be divided into two categories. The first group of algorithms finds representatives from data that lie in one or multiple low-dimensional subspaces and typically operate on the measurement data vectors directly [5, 6, 7, 8, 9, 10, 11]. The Rank Revealing QR (RRQR) algorithm [6, 9] assumes that the data come from a low-rank model and tries to find a subset of columns of the data matrix that corresponds to the best conditioned submatrix. Randomized and greedy algorithms have also been proposed to find a subset of the columns of a low-rank matrix [5, 8, 10]. Assuming that the data can be expressed as a linear combination of the representatives, [7, 11] formulate the problem of finding representatives as a joint-sparse recovery problem, [7] showing that when the data lie in a union of low-rank models, the algorithm finds representatives from each low-rank model.

The second group of algorithms finds representatives by assuming that there is a natural grouping of the data collection based on an appropriate measure of similarity between pairs of data points [2, 4, 12, 13, 14]. As a result, such algorithms typically operate on similarities/dissimilarities between data points. The Kmedoids algorithm [2] tries to find $K$ representatives from pairwise dissimilarities between data points. As solving the original optimization program is, in general, NP-hard [12], an iterative approach is employed. The performance of Kmedoids, similar to Kmeans [15], depends on initialization and decreases as the number of representatives, $K$, increases. The Affinity Propagation (AP) algorithm [4, 13, 14] tries to find representatives from pairwise similarities between data points by using a message passing algorithm. While AP has suboptimal properties and finds approximate solutions, it does not require initialization and has been shown to perform well in problems such as unsupervised image categorization [16] and facility location problems [17].

In this paper, we propose an algorithm for selecting representatives of a data collection given dissimilarities between pairs of data points. We propose a row-sparsity regularized [18, 19] trace minimization program whose objective is to find *a few representatives* that *encode well* the collection of data points according to the provided dissimilarities. The solution of the proposed optimization program finds the representatives and the probability that each data point is associated with each one of the representatives. Instead of choosing the number of representatives, the regularization parameter puts a trade-off between the number of representatives and the encoding cost of the data points via the representatives based on the dissimilarities. We obtain the range of the regularization parameter where the solution of the proposed optimization program changes from selecting one representative for all data points to selecting each data point as a representative. When there is a clustering of data points, defined based on their dissimilarities, we show that, for a suitable range of the regularization parameter, the algorithm finds representatives from each cluster. Moreover, data points in each cluster select representatives only from the same cluster. Unlike metric-based methods, we do not require that the dissimilarities come from a metric. Specifically, the dissimilarities can be asymmetric or can violate the triangle inequality. We demonstrate the effectiveness of the proposed algorithm on synthetic data and real-world image and text data.

## 2   Problem Statement

We consider the problem of finding representatives from a collection of $N$ data points. Assume we are given a set of nonnegative dissimilarities $\{d_{ij}\}_{i,j=1,\ldots,N}$ between every pair of data points $i$ and $j$. The dissimilarity $d_{ij}$ indicates how well the data point $i$ is suited to be a representative of the data point $j$. More specifically, the smaller the value of $d_{ij}$ is, the better the data point $i$ is a representative of the data point $j$.[1] Such dissimilarities can be built from measured data points, e.g., by using the Euclidean/geodesic distances or the inner products between data points. Dissimilarities can also be given directly without accessing or measuring the data points, e.g., they can be subjective measurements of the relationships between different objects. We can arrange the dissimilarities into a matrix of the form

$$\boldsymbol{D} \triangleq \begin{bmatrix} \boldsymbol{d}_1^\top \\ \vdots \\ \boldsymbol{d}_N^\top \end{bmatrix} = \begin{bmatrix} d_{11} & d_{12} & \cdots & d_{1N} \\ \vdots & \vdots & & \vdots \\ d_{N1} & d_{N2} & \cdots & d_{NN} \end{bmatrix} \in \mathbb{R}^{N \times N}, \tag{1}$$

where $\boldsymbol{d}_i \in \mathbb{R}^N$ denotes the $i$-th row of $\boldsymbol{D}$.

**Remark 1** *We do not require the dissimilarities to satisfy the triangle inequality. In addition, we do not assume symmetry on the pairwise dissimilarities. $\boldsymbol{D}$ can be asymmetric, where $d_{ij} \neq d_{ji}$ for some pairs of data points. In other words, how well data point $i$ represents data point $j$ can be different from how well $j$ represents $i$. In the experiments, we will show an example of asymmetric dissimilarities for finding representative sentences in text documents.*

Given $\boldsymbol{D}$, our goal is to select a subset of data points, called *representatives* or *exemplars*, that efficiently represent the collection of data points. We consider an optimization program that promotes selecting a few data points that can well encode all data points via the dissimilarities. To do so, we consider variables $z_{ij}$ associated with dissimilarities $d_{ij}$ and denote by the matrix of all variables as

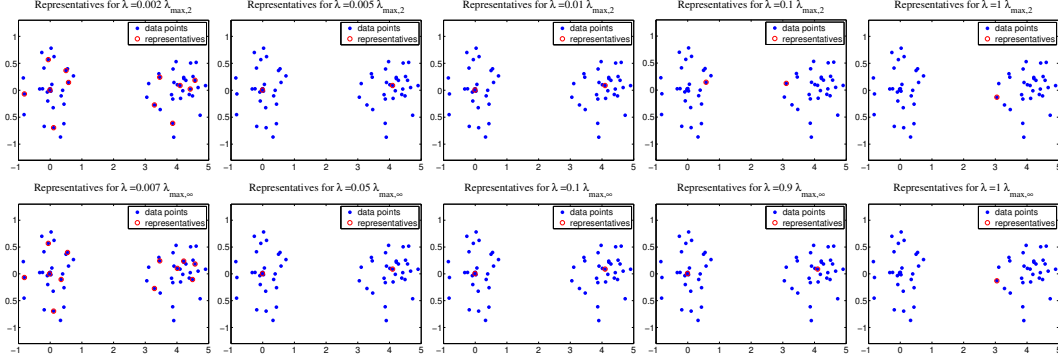

Figure 1: Data points (blue dots) in two clusters and the representatives (red circles) found by the proposed optimization program in (4) for several values of $\lambda$ with $\lambda_{\mathrm{max},q}$ defined in (6). Top: $q = 2$, Bottom: $q = \infty$.

$$
\boldsymbol{Z} \triangleq \begin{bmatrix} \boldsymbol{z}_1^\top \\ \vdots \\ \boldsymbol{z}_N^\top \end{bmatrix} = \begin{bmatrix} z_{11} & z_{12} & \cdots & z_{1N} \\ \vdots & \vdots & & \vdots \\ z_{N1} & z_{N2} & \cdots & z_{NN} \end{bmatrix} \in \mathbb{R}^{N \times N}, \tag{2}
$$

where $\boldsymbol{z}_i \in \mathbb{R}^N$ denotes the $i$-th row of $\boldsymbol{Z}$. We interpret $z_{ij}$ as the probability that data point $i$ be a representative for data point $j$, hence $z_{ij} \in [0,1]$. A data point $j$ can have multiple representatives in which case $z_{ij} > 0$ for all the indices $i$ of the representatives. As a result, we must have $\sum_{i=1}^N z_{ij} = 1$, which ensures that the total probability of data point $j$ choosing all its representatives is equal to one. Our goal is to select *a few* representatives that *well encode* the data collection according to the dissimilarities. To do so, we propose a row-sparsity regularized trace minimization program on $\boldsymbol{Z}$ that consists of two terms. First, we want the representatives to encode well all data points via dissimilarities. If the data point $i$ is chosen to be a representative of a data point $j$ with probability $z_{ij}$, the cost of encoding $j$ with $i$ is $d_{ij} z_{ij} \in [0, d_{ij}]$. Hence, the total cost of encoding $j$ using all its representatives is $\sum_{i=1}^N d_{ij} z_{ij}$. Second, we would like to have as few representatives as possible for all the data points. When the data point $i$ is a representative of some of the data points, we have $\boldsymbol{z}_i \neq \boldsymbol{0}$, i.e., the $i$-th row of $\boldsymbol{Z}$ is nonzero. Having a few representatives then corresponds to having a few nonzero rows in the matrix $\boldsymbol{Z}$. Putting these two goals together, we consider the following minimization program

$$
\min \ \sum_{j=1}^N \sum_{i=1}^N d_{ij} z_{ij} + \lambda \sum_{i=1}^N \mathrm{I}(\|\boldsymbol{z}_i\|_q) \quad \text{s.t.} \quad z_{ij} \geq 0, \ \forall i,j; \ \sum_{i=1}^N z_{ij} = 1, \ \forall j, \tag{3}
$$

where $\mathrm{I}(\cdot)$ denotes the indicator function, which is zero when its argument is zero and is one otherwise. The first term in the objective function corresponds to the total cost of encoding all data points using the representatives and the second term corresponds to the cost associated with the number of the representatives. The parameter $\lambda > 0$ sets the trade-off between the two terms. Since the minimization in (3) that involves counting the number of nonzero rows of $\boldsymbol{Z}$ is, in general, NP-hard, we consider the following standard convex relaxation

$$
\min \ \sum_{j=1}^N \sum_{i=1}^N d_{ij} z_{ij} + \lambda \sum_{i=1}^N \|\boldsymbol{z}_i\|_q \quad \text{s.t.} \quad z_{ij} \geq 0, \ \forall i,j; \ \sum_{i=1}^N z_{ij} = 1, \ \forall j, \tag{4}
$$

where, instead of counting the number of nonzero rows of $\boldsymbol{Z}$, we use the sum of the $\ell_q$-norms of the rows of $\boldsymbol{Z}$. Typically, we choose $q \in \{2, \infty\}$ for which the optimization program (4) is convex.[2] Note that the optimization program (4) can be rewritten in the matrix form as

$$
\min \ \mathrm{tr}(\boldsymbol{D}^\top \boldsymbol{Z}) + \lambda \|\boldsymbol{Z}\|_{1,q} \quad \text{s.t.} \quad \boldsymbol{Z} \geq \boldsymbol{0}, \ \boldsymbol{1}^\top \boldsymbol{Z} = \boldsymbol{1}^\top, \tag{5}
$$

where $\mathrm{tr}(\cdot)$ denotes the trace operator, $\|\boldsymbol{Z}\|_{1,q} \triangleq \sum_{i=1}^N \|\boldsymbol{z}_i\|_q$, and $\boldsymbol{1}$ denotes an $N$-dimensional vector whose elements are all equal to one.

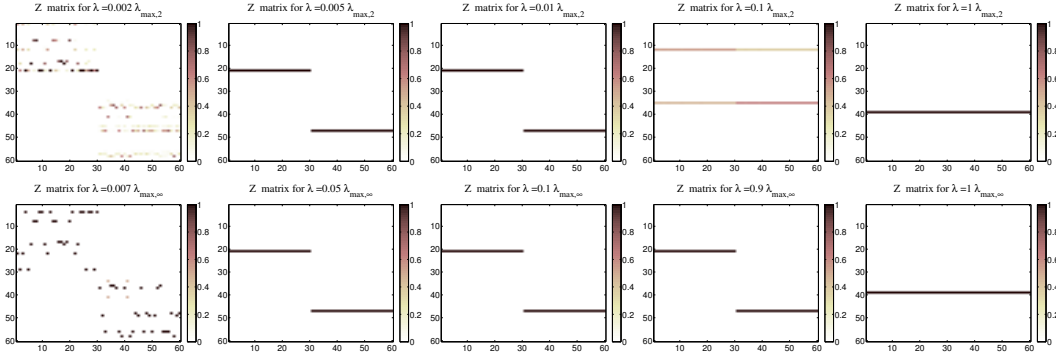

Figure 2: For the data points shown in Fig. 1, the matrix $\boldsymbol{Z}$ obtained by the proposed optimization program in (4) is shown for several values of $\lambda$, where $\lambda_{\mathrm{max},q}$ is defined in (6). Top: $q = 2$, Bottom: $q = \infty$.

As we change the regularization parameter $\lambda$ in (4), the number of representatives found by the algorithm changes. For small values of $\lambda$, where we put more emphasis on better encoding data points via representatives, we obtain more representatives. In the limiting case of $\lambda \to 0$ all points are selected as representatives, each point being the representative of itself, i.e., $\boldsymbol{z}_{ii} = 1$ for all $i$. On the other hand, for large values of $\lambda$, where we put more emphasis on the row-sparsity of $\boldsymbol{Z}$, we select a small number of representatives. In the limiting case of $\lambda \to \infty$, we select only one representative for all data points. Figures 1 and 2 illustrate the representatives and the matrix $\boldsymbol{Z}$, respectively, for several values of $\lambda$. In Section 3, we compute the range of $\lambda$ for which the solution of (4) changes from a single representative to all points being representatives. Note that, similar to the relationship between sparse dictionary leaning [20] and Kmeans, there is a relationship between our method and Kmedoids. A discussion of this is part of a future publication.

Once we have solved the optimization program (4), we can find the representative indices from the nonzero rows of $\boldsymbol{Z}$. We can also obtain the clustering of data points into $K$ clusters associated with $K$ representatives by assigning each data point to its closets representative. More specifically, if $i_1, \cdots, i_K$ denote the indices of the representatives, data point $j$ is assigned to the representative $R(j)$ according to $R(j) = \operatorname{argmin}_{\ell \in \{i_1, \cdots, i_K\}} d_{\ell j}$. As mentioned before, the solution $\boldsymbol{Z}$ gives the probability that each data point is associated with each one of the representatives, which also provides a soft clustering of data points to the representatives. In Section 3 we show that when there is a clustering of data points based on their dissimilarities (see Definition 1), each point selects representatives from its own cluster.

## 3 Theoretical Analysis

In this section, we consider the optimization program (4) and study the behavior of its solution as a function of the regularization parameter. First, we analyze the solution of (4) for a sufficiently large value of $\lambda$. We obtain a threshold value on $\lambda$ after which the solution of (4) remains the same, selecting only one representative data point. More specifically, we show the following result.

**Theorem 1** *Consider the optimization program* (4). *Let* $\ell \triangleq \operatorname{argmin}_i \mathbf{1}^\top \boldsymbol{d}_i$ *and*

$$\lambda_{\mathrm{max},2} \triangleq \max_{i \neq \ell} \frac{\sqrt{N}}{2} \cdot \frac{\|\boldsymbol{d}_i - \boldsymbol{d}_\ell\|_2^2}{\mathbf{1}^\top (\boldsymbol{d}_i - \boldsymbol{d}_\ell)}, \qquad \lambda_{\mathrm{max},\infty} \triangleq \max_{i \neq \ell} \frac{\|\boldsymbol{d}_i - \boldsymbol{d}_\ell\|_1}{2}. \tag{6}$$

*For* $q \in \{2, \infty\}$, *when* $\lambda \geq \lambda_{\mathrm{max},q}$, *the solution of the optimization program* (4) *is equal to* $\boldsymbol{Z} = \boldsymbol{e}_\ell \mathbf{1}^\top$, *where* $\boldsymbol{e}_\ell$ *denotes the vector whose elements are all zero except its $\ell$-th element, which is equal to 1. In other words, the solution of* (4) *for* $\lambda \geq \lambda_{\mathrm{max},q}$ *corresponds to choosing only the $\ell$-th data point as the representative of all the data points.*

Note that the threshold value of the regularization parameter, for which we obtain only one representative, is different for $q = 2$ and $q = \infty$. However, the two cases obtain the same representative given by the data point for which $\mathbf{1}^\top \boldsymbol{d}_i$ is minimum, i.e., the data point with the smallest sum of

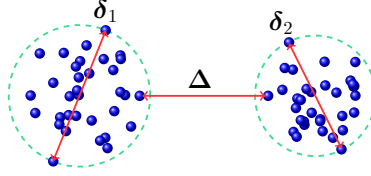

Figure 3: Data points in two clusters with dissimilarities given by pairwise Euclidean distances. For $\lambda < \Delta - \max\{\delta_1, \delta_2\}$, in the solution of the optimization program (4), points in each cluster are represented by representatives from the same cluster.

dissimilarities to other data points. Notice also that when the dissimilarities are the Euclidean distances between the data points, the single representative corresponds to the data point closest to the geometric median of all data points, as shown in the right plot of Figure 1.

When the regularization parameter $\lambda$ is smaller than the threshold in (6), the optimization program in (4) can find multiple representatives for each data point. However, when there is a clustering of data points based on their dissimilarities (see Definition 1), we expect to select representatives from each cluster. In addition, we expect that the data points in each cluster be associated with the representatives in that cluster only.

**Definition 1** *Given dissimilarities $\{d_{ij}\}_{i,j=1,\ldots,N}$ between $N$ data points, we say that the data partitions into $n$ clusters $\{\mathcal{C}_i\}_{i=1}^n$ according to the dissimilarities, if for any data point $j'$ in any $\mathcal{C}_j$, the largest dissimilarity to other data points in $\mathcal{C}_j$ is strictly smaller than the smallest dissimilarity to the data points in any $\mathcal{C}_i$ different from $\mathcal{C}_j$, i.e.,*

$$\max_{i' \in \mathcal{C}_j} d_{i'j'} < \min_{i \neq j} \min_{i' \in \mathcal{C}_i} d_{i'j'}, \quad \forall j = 1, \ldots, n, \ \forall j' \in \mathcal{C}_j. \tag{7}$$

*In other words, the data partitions into clusters $\{\mathcal{C}_i\}_{i=1}^n$, when the interclass dissimilarity is smaller than the intraclass dissimilarity.*

Next, we show that for a suitable range of the regularization parameter that depends on the intraclass and interclass dissimilarities, the probability that a point chooses representatives from other clusters is zero. More precisely, we have the following result.

**Theorem 2** *Given dissimilarities $\{d_{ij}\}_{i,j=1,\ldots,N}$ between $N$ data points, assume that the data partitions into $n$ clusters $\{\mathcal{C}_i\}_{i=1}^n$ according to Definition 1. Let $\lambda_c$ be defined as*

$$\lambda_c \triangleq \min_j \min_{j' \in \mathcal{C}_j} (\min_{i \neq j} \min_{i' \in \mathcal{C}_i} d_{i'j'} - \max_{i' \in \mathcal{C}_j} d_{i'j'}). \tag{8}$$

*Then for $\lambda \leq \lambda_c$, the optimization program (4) finds representatives in each cluster, where the data points in every $\mathcal{C}_i$ select representatives only from $\mathcal{C}_i$. A less tight clustering threshold $\lambda'_c \leq \lambda_c$ on the regularization parameter is given by*

$$\lambda'_c \triangleq \min_{i \neq j} \min_{i' \in \mathcal{C}_i, j' \in \mathcal{C}_j} d_{i'j'} - \max_i \max_{i' \neq j' \in \mathcal{C}_i} d_{i'j'}. \tag{9}$$

The first term in the right-hand-side of (9) shows the minimum dissimilarity between data points in two different clusters. The second term in the right-hand-side of (9) shows the maximum, over all clusters, of the dissimilarity between different data points in each cluster. When $\lambda_c$ or $\lambda'_c$ increase, e.g., when the intraclass dissimilarities increase or the interclass dissimilarities decrease, the maximum possible $\lambda$ for which we obtain clustering increases. As an illustrative example, consider Figure 3, where data points are distributed in two clusters according to the dissimilarities given by the pairwise Euclidean distances of the data points. Let $\delta_i$ denote the diameter of cluster $i$ and $\Delta$ be the minimum distance among pairs of data points in different clusters. Assuming $\max\{\delta_1, \delta_2\} < \Delta$, for $\lambda < \Delta - \max\{\delta_1, \delta_2\}$, the solution of the optimization program (4) is of the form $\boldsymbol{Z} = \boldsymbol{\Gamma} \begin{bmatrix} \boldsymbol{Z}_1 & \boldsymbol{0} \\ \boldsymbol{0} & \boldsymbol{Z}_2 \end{bmatrix}$, where $\boldsymbol{\Gamma} \in \mathbb{R}^{N \times N}$ is a permutation matrix corresponding to the separation of the data into the two clusters.

**Remark 2** *The results of Theorems 1 and 2 suggest that there is a range of the regularization parameter for which we obtain only one representative from each cluster. In other words, if*

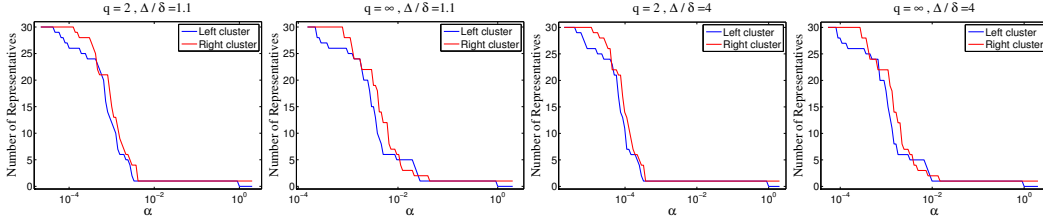

Figure 4: Number of representatives obtained by the proposed optimization program in (4) for data points in the two clusters shown in Fig. 1 as a function of the regularization parameter $\lambda = \alpha \lambda_{\max,q}$ with $q \in \{2, \infty\}$.

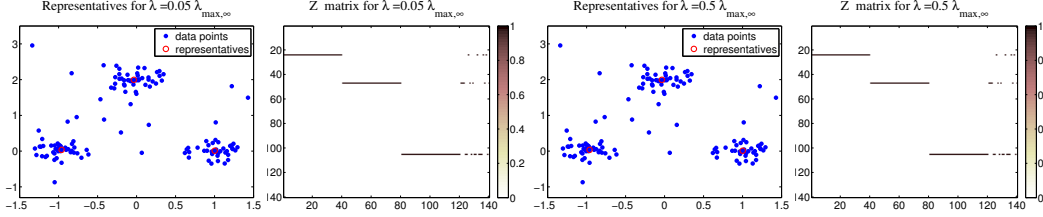

Figure 5: Representatives and the probability matrix $\mathbf{Z}$ obtained by our proposed algorithm in (4) for $q = \infty$. 20 random data points are added to 120 data points generated by a mixture of 3 Gaussian distributions.

$\lambda_{\max,q}(\mathcal{C}_i)$ *denotes the threshold on* $\lambda$ *after which we obtain only one representative from* $\mathcal{C}_i$*, then for* $\max_i \lambda_{\max,q}(\mathcal{C}_i) \leq \lambda < \lambda_c$*, the data points in each* $\mathcal{C}_i$ *select only one representative that is in* $\mathcal{C}_i$*. As we will show in the experiments, such an interval often exists and can, in fact, be large.*

For a sufficiently small value of $\lambda$, where we put less emphasis in the row-sparsity term in the optimization program (4), each data point becomes a representative, i.e., $z_{ii} = 1$ for all $i$. In such a case, each data point forms its own cluster. From the result in Theorem 2, we obtain a threshold $\lambda_{\min}$ such that for $\lambda \leq \lambda_{\min}$ the solution $\mathbf{Z}$ is equal to the identity matrix.

**Corollary 1** *Let* $\lambda_{\min,q} \triangleq \min_j(\min_{i \neq j} d_{ij} - d_{jj})$ *for* $q \in \{2, \infty\}$*. For* $\lambda \leq \lambda_{\min,q}$*, the solution of the optimization program* (4) *for* $q \in \{2, \infty\}$ *is equal to the identity matrix. In other words, each data point is the representative of itself.*

# 4 Experiments

In this section, we evaluate the performance of the proposed algorithm on synthetic and real datasets. As scaling of $\mathbf{D}$ and $\lambda$ by the same value does not change the solution of (4), we always scale dissimilarities to lie in $[0, 1]$ by dividing the elements of $\mathbf{D}$ by its largest element. Unless stated otherwise, we typically set $\lambda = \alpha \lambda_{\max,q}$ with $\alpha \in [0.01, 0.1]$, for which we obtain good results.

## 4.1 Experiments on Synthetic Data

We consider the synthetic dataset shown in Figure 1 that consists of data points distributed around two clusters. We run the proposed optimization program in (4) for both $q = 2$ and $q = \infty$ for several values of $\lambda$. Figures 1 and 2 show the representatives and the matrix of variables $\mathbf{Z}$, respectively, for several values of the regularization parameter. Notice that, as discussed before, for small values of $\lambda$, we obtain more representatives and as we increase $\lambda$, the number of representatives decreases. When the regularization parameter reaches $\lambda_{\max,q}$, computed using our theoretical analysis, we obtain only one representative for the dataset. It is important to note that, as we showed in the theoretical analysis, when the regularization parameter is sufficiently small, data points in each cluster only select representatives from that cluster (see Figure 2), i.e., $\mathbf{Z}$ has a block-diagonal structure when its columns are permuted according to the clusters. Moreover, as Figure 2 shows, for a sufficiently large range of the regularization parameter, we obtain only one representative from each cluster. To better see this, we run the optimization program with $\lambda = \alpha \lambda_{\max,q}$ for different values of $\alpha$. The two left-hand side plots in Figure 4 show the number of the representatives for $q = 2$ and $q = \infty$, respectively, from each of the two clusters.

As shown, when $\lambda$ gets larger than $\lambda_{\max,q}$, we obtain only one representative from the right cluster and no representative from the left cluster, i.e., as expected, we obtain one representative for all the data points. Also, when $\lambda$ gets smaller than $\lambda_{\min,q}$, all data points become representatives, as

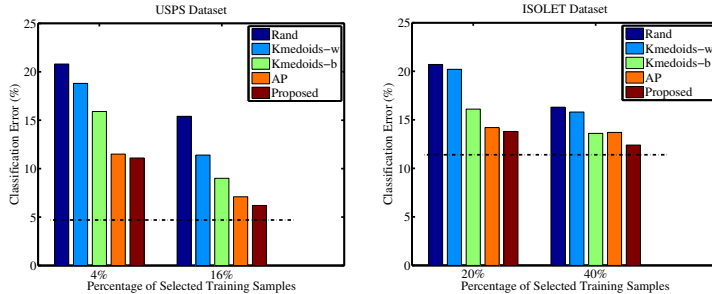

Figure 6: Classification error on the USPS (left) and ISOLET (right) datasets using representatives obtained by different algorithms. Horizontal axis shows the percentage of the selected representatives from each class (averaged over all classes). Dashed line shows the classification error (%) using all the training samples.

expected from our theoretical result. It is also important to note that, for a sufficiently large range of the values of $\lambda$, we select only one representative from each cluster. The two right-hand side plots in Figure 4 show the number of the representatives when we increase the distance between the two clusters. Notice that we obtain similar results as before except that the range of $\lambda$ for which we select one representative from each cluster has increased. This is also expected from our theoretical analysis, since $\lambda_c$ in (8) increases as the distance between the two clusters increases.

Note that we also obtain similar results for larger number of clusters. For better visualization, we have shown the results for only two clusters. Also, when there is not a clear partitioning of the data points into clusters according to Definition 1, e.g., when there are data points distributed between different clusters, as shown in Figure 5, we still obtain similar results to what we have discussed in our theoretical analysis. This suggests the existence of stronger theoretical guarantees for our proposed algorithm, which is the subject of our future work.

## 4.2 Experiments on Real Data

In this section, we evaluate the performance of our proposed algorithm on real image and text data. We report the result for $q = \infty$ as it typically obtains better results than $q = 2$.

### 4.2.1 NN Classification using Representatives

First, we consider the problem of finding prototypes for classification using the nearest neighbor (NN) algorithm [15]. Finding representatives that correspond to the modes of the data distribution helps to significantly reduce the computational cost and memory requirements of classification algorithms, while maintaining their performance. To investigate the effectiveness of our proposed method for finding informative prototypes for classification, we consider two datasets of USPS [21] and ISOLET [22]. We find the representatives of the training data in each class of a dataset and use the representatives as a reduced training set to perform NN classification on the test data. We obtain the representatives by taking dissimilarities to be pairwise Euclidean distances between data points. We compare our proposed algorithm with AP [4], Kmedoids [2], and random selection of data points (Rand) as the baseline. Since Kmedoids depends on initialization, we run the algorithm 1000 times with different random initializations and report the results corresponding to the best solution (lowest energy) and the worst solution (highest energy) as Kmedoids-w and Kmedoids-b, respectively. To have a fair comparison, we run all algorithms so that they obtain the same number of representatives.

Figure 6 shows the average classification errors using the NN method for the two datasets. The classification error using all training samples of each dataset is also shown with a black dashed line. As the results show, the classification performance using the representatives found by our proposed algorithm is close to that of using all the training samples. Specifically, in the USPS dataset, using representatives found by our proposed method, which consist of only $16\%$ of the training samples, we obtain $6.2\%$ classification error compared to $4.7\%$ error obtained using all the training samples. In the ISOLET dataset, with representatives corresponding to less than half of the training samples, we obtain very close classification performance to using all the training samples ($12.4\%$ error compared to $11.4\%$ error). Notice that when the number of representatives decreases, as expected, the classification performance also decreases. However, in all cases, our proposed algorithm as well as AP are less affected by the decrease in the number of the representatives.

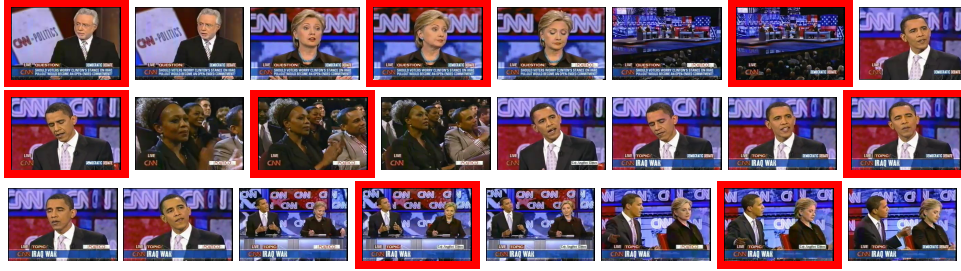

Figure 7: Some frames of a political debate video, which consists of multiple shots, and the automatically computed representatives (inside red rectangles) of the whole video sequence using our proposed algorithm.

### 4.2.2 Video Summarization using Representatives

We now evaluate our proposed algorithm for finding representative frames of video sequences. We take a political debate video [7], downsample the frames to $80 \times 100$ pixels, and convert each frame to a grayscale image. Each data point then corresponds to an 8000-dimensional vector obtained by vectorizing each grayscale downsampled frame. We set the dissimilarities to be the Euclidean distances between pairs of data points. Figure 7 shows some frames of the video and the representatives computed by our method. Notice that we obtain a representative for each shot of the video. It is worth mentioning that the computed representatives do not change for $\lambda \in [2.68, 6.55]$.

### 4.2.3 Finding Representative Sentences in Text Documents

As we discussed earlier, our proposed algorithm can deal with dissimilarities that are not necessarily metric, i.e., can be asymmetric or violate the triangle inequality. We consider now an example of asymmetric dissimilarities where we find representative sentences in the text document of this paper. We compute the dissimilarities between sentences using an information theory-based criterion as follows [4]: we treat each sentence as a "bag of words" and compute $d_{ij}$ (how well sentence $i$ represents sentence $j$) based on the sum of the costs of encoding every word in sentence $j$ using the words in sentence $i$. More precisely, for sentences in the text of the paper, we extract the words delimited by spaces, we remove all punctuations, and eliminate words that have less than 5 characters. For each word in sentence $j$, if the word matches[3] a word in sentence $i$, we set the encoding cost for the word to the logarithm of the number of words in sentence $i$, which is the cost of encoding the index of the matched word. Otherwise, we set the encoding cost for the word to the logarithm of the number of the words in the text dictionary, which is the cost of encoding the index of the word in all the text. We also compute $d_{ii}$ using the same procedure, i.e., $d_{ii} \neq 0$, which penalizes selecting very long sentences. We found that $96\%$ of the dissimilarities are asymmetric. The four representative sentences obtained by our algorithm summarize the paper as follows:

–Given pairwise dissimilarities between data points, we consider the problem of finding a subset of data points, called representatives or exemplars, that can efficiently describe the data collection.

–We obtain the range of the regularization parameter for which the solution of the proposed optimization program changes from selecting one representative for all data points to selecting all data points as representatives.

–When there is a clustering of data points, defined based on their dissimilarities, we show that, for a suitable range of the regularization parameter, the algorithm finds representatives from each cluster.

–As the results show, the classification performance using the representatives found by our proposed algorithm is close to that of using all the training samples.

## Acknowledgment

E. Elhamifar and R. Vidal are supported by grants NSF CNS-0931805, NSF ECCS-0941463, NSF OIA-0941362, and ONR N00014-09-10839. G. Sapiro acknowledges partial support by ONR, DARPA, NSF, NGA, and AFOSR grants.

## Footnotes

[1]$d_{ii}$ can be set to have a nonzero value, as we will show in the experiments on the text data.

[2]It is typically the case that $q = \infty$ favors having 0 and 1 elements for $\boldsymbol{Z}$, while $q = 2$ allows elements that more often take other values in $[0, 1]$. Note that $q = 1$ also imposes *sparsity* in the nonzero *rows* of $\boldsymbol{Z}$, which is not desirable since it promotes only a few data points to be associated with each representative.

[3]We consider a word to match another word, if either word is a substring of the other.

# References

[1] S. Garcia, J. Derrac, J. R. Cano, and F. Herrera, "Prototype selection for nearest neighbor classification: Taxonomy and empirical study," *IEEE Transactions on Pattern Analysis and Machine Intelligence*, vol. 34, no. 3, pp. 417–435, 2012.

[2] L. Kaufman and P. Rousseeuw, "Clustering by means of medoids," *In Y. Dodge (Ed.), Statistical Data Analysis based on the L1 Norm (North-Holland, Amsterdam)*, pp. 405–416, 1987.

[3] M. Gu and S. C. Eisenstat, "Efficient algorithms for computing a strong rank-revealing qr factorization," *SIAM Journal on Scientific Computing*, vol. 17, pp. 848–869, 1996.

[4] B. J. Frey and D. Dueck, "Clustering by passing messages between data points," *Science*, vol. 315, pp. 972–976, 2007.

[5] J. A. Tropp, "Column subset selection, matrix factorization, and eigenvalue optimization," *ACM-SIAM Symp. Discrete Algorithms (SODA)*, pp. 978–986, 2009.

[6] C. Boutsidis, M. W. Mahoney, and P. Drineas, "An improved approximation algorithm for the column subset selection problem," in *Proceedings of SODA*, 2009, pp. 968–977.

[7] E. Elhamifar, G. Sapiro, and R. Vidal, "See all by looking at a few: Sparse modeling for finding representative objects," in *IEEE Conference on Computer Vision and Pattern Recognition*, 2012.

[8] J. Bien, Y. Xu, and M. W. Mahoney, "CUR from a sparse optimization viewpoint," *NIPS*, 2010.

[9] T. Chan, "Rank revealing QR factorizations," *Lin. Alg. and its Appl.*, vol. 88-89, pp. 67–82, 1987.

[10] L. Balzano, R. Nowak, and W. Bajwa, "Column subset selection with missing data," in *NIPS Workshop on Low-Rank Methods for Large-Scale Machine Learning*, 2010.

[11] E. Esser, M. Moller, S. Osher, G. Sapiro, and J. Xin, "A convex model for non-negative matrix factorization and dimensionality reduction on physical space," *IEEE Transactions on Image Processing*, vol. 21, no. 7, pp. 3239–3252, 2012.

[12] M. Charikar, S. Guha, A. Tardos, and D. B. Shmoys, "A constant-factor approximation algorithm for the k-median problem," *Journal of Computer System Sciences*, vol. 65, no. 1, pp. 129–149, 2002.

[13] B. J. Frey and D. Dueck, "Mixture modeling by affinity propagation," *Neural Information Processing Systems*, 2006.

[14] I. E. Givoni, C. Chung, and B. J. Frey, "Hierarchical affinity propagation," *Conference on Uncertainty in Artificial Intelligence*, 2011.

[15] R. Duda, P. Hart, and D. Stork, *Pattern Classification*. Wiley-Interscience, October 2004.

[16] D. Dueck and B. J. Frey, "Non-metric affinity propagation for unsupervised image categorization," *International Conference in Computer Vision*, 2007.

[17] N. Lazic, B. J. Frey, and P. Aarabi, "Solving the uncapacitated facility location problem using message passing algorithms," *International Conference on Artificial Intelligence and Statistics*, 2007.

[18] R. Jenatton, J. Y. Audibert, and F. Bach, "Structured variable selection with sparsity-inducing norms," *Journal of Machine Learning Research*, vol. 12, pp. 2777–2824, 2011.

[19] J. A. Tropp., "Algorithms for simultaneous sparse approximation. part ii: Convex relaxation," *Signal Processing, special issue "Sparse approximations in signal and image processing"*, vol. 86, pp. 589–602, 2006.

[20] M. Aharon, M. Elad, and A. M. Bruckstein, "K-SVD: An algorithm for designing overcomplete dictionaries for sparse representation," *IEEE Trans. on Signal Processing*, vol. 54, no. 11, pp. 4311–4322, 2006.

[21] J. J. Hull, "A database for handwritten text recognition research," *IEEE Transactions on Pattern Analysis and Machine Intelligence*, vol. 16, no. 5, pp. 550–554, 1994.

[22] M. Fanty and R. Cole, "Spoken letter recognition," in *Neural Information Processing Systems*, 1991.

